# Evidence Optimization Techniques for Estimating Stimulus-Response Functions

**Maneesh Sahani**
Gatsby Unit, UCL
17 Queen Sq., London, WC1N 3AR, UK.
maneesh@gatsby.ucl.ac.uk

**Jennifer F. Linden**
Keck Center, UCSF
San Francisco, CA 94143–0732, USA.
linden@phy.ucsf.edu

## Abstract

An essential step in understanding the function of sensory nervous systems is to characterize as accurately as possible the stimulus-response function (SRF) of the neurons that relay and process sensory information. One increasingly common experimental approach is to present a rapidly varying complex stimulus to the animal while recording the responses of one or more neurons, and then to directly estimate a functional transformation of the input that accounts for the neuronal firing. The estimation techniques usually employed, such as Wiener filtering or other correlation-based estimation of the Wiener or Volterra kernels, are equivalent to maximum likelihood estimation in a Gaussian-output-noise regression model. We explore the use of Bayesian evidence-optimization techniques to condition these estimates. We show that by learning hyperparameters that control the smoothness and sparsity of the transfer function it is possible to improve dramatically the quality of SRF estimates, as measured by their success in predicting responses to novel input.

## 1 Introduction

A common experimental approach to the measurement of the stimulus-response function (SRF) of sensory neurons, particularly in the visual and auditory modalities, is "reverse correlation" and its related non-linear extensions [1]. The neural response $y(t)$ to a continuous, rapidly varying stimulus $x(t)$, is measured and used in an attempt to reconstruct the functional mapping $y(t) = F[x(t)]$. In the simplest case, the functional is taken to be a finite impulse response (FIR) linear filter; if the input is white the filter is identified by the spike-triggered average of the stimulus, and otherwise by the Wiener filter. Such linear filter estimates are often called STRFs for spatio-temporal (in the visual case) or spectro-temporal (in the auditory case) receptive fields. The general the SRF may also be parameterized on the basis of known or guessed non-linear properties of the neurons, or may be expanded in terms of the Volterra or Wiener integral power series. In the case of the Wiener expansion, the integral kernels are usually estimated by measuring various cross-moments of $y(t)$ and $x(t)$.

In practice, the stimulus is often a discrete-time process $\{x_t\}$. In visual experiments, the discretization may correspond to the frame rate of the display. In the auditory experiments that will be considered below, it is set by the rate of the component tone pulses in a random

chord stimulus. On time-scales finer than that set by this discretization rate, the stimulus is strongly autocorrelated. This makes estimation of the SRF at a finer time-scale extremely non-robust. We therefore lose very little generality by discretizing the response with the same time-step, obtaining a response histogram $\{y_t\}$.

In this discrete-time framework, the estimation of FIR Wiener-Volterra kernels (of any order) corresponds to linear regression. To estimate the first-order kernel up to a given maximum time lag $L$, we construct a set of input lag-vectors $\{\mathbf{x}_t = (x_\tau)_{\tau = t-L+1\ldots t}\}$. If a single stimulus frame, $x_t$, is itself a $D$-dimensional vector (representing, say, pixels in an image or power in different frequency bands) then the lag vectors are formed by concatenating $L$ stimulus frames together into vectors of length $LD$. The Wiener filter is then obtained by least-squares linear regression from the lag vectors $\{\mathbf{x}_t\}$ to the corresponding observed activities $\{y_t\}$.

Higher-order kernels can also be found by linear regression, using augmented versions of the stimulus lag vectors. For example, the second-order kernel is obtained by regression using input vectors formed by all quadratic combinations of the elements of $\mathbf{x}_t$ (or, equivalently, by support-vector-like kernel regression using a homogeneous second-order polynomial kernel). The present paper will be confined to a treatment of the linear case. It should be clear, however, that the basic techniques can be extended to higher orders at the expense of additional computational load, provided only that a sensible definition of smoothness in these higher-order kernels is available.

The least-squares solution to a regression problem is identical to the maximum likelihood (ML) value of the weight vector $\mathbf{w}$ for the probabilistic regression model with Gaussian output noise of constant variance $\sigma^2$:

$$ y_t \mid \mathbf{x}_t \sim \mathcal{N}(\mathbf{w}^\mathsf{T}\mathbf{x}_t, \sigma^2). \tag{1} $$

As is common with ML learning, weight vectors obtained in this way are often overfit to the training data, and so give poor estimates of the true underlying stimulus-response function. This is the case even for linear models. If the stimulus is uncorrelated, the ML-estimated weight along some input dimension is proportional to the observed correlation between that dimension of the stimulus and the output response. Noise in the output can introduce spurious input-output correlations and thus result in erroneous weight values. Furthermore, if the true relationship between stimulus and response is non-linear, limited sampling of the input space may also lead to observed correlations that would have been absent given unlimited data.

The statistics and machine learning literatures provide a number of techniques for the containment of overfitting in probabilistic models. Many of these approaches are equivalent to the maximum *a posteriori* (MAP) estimation of parameters under a suitable prior distribution. Here, we investigate an approach in which these prior distributions are optimized with reference to the data; as such, they cease to be "prior" in a strict sense, and instead become part of a hierarchical probabalistic model. A distribution on the regression parameters is first specified up to the unknown values of some hyperparameters. These hyperparameters are then adjusted so as to maximize the marginal likelihood or "evidence" — that is, the probability of the data given the hyperparameters, with the parameters themselves integrated out. Finally, the estimate of the parameters is given by the MAP weight vector under the optimized "prior". Such evidence optimization schemes have previously been used in the context of linear, kernel and Gaussian-process regression. We show that, with realistic data volumes, such techniques provide considerably better estimates of the stimulus-response function than do the unregularized (ML) Wiener estimates.

## 2 Test data and methods

A diagnostic of overfitting, and therefore divergence from the true stimulus-response relationship, is that the resultant model generalizes poorly; that is, it does not predict actual responses to novel stimuli well. We assessed the generalization ability of parameters chosen by maximum likelihood and by various evidence optimization schemes on a set of responses collected from the auditory cortex of rodents. As will be seen, evidence optimization yielded estimates that generalized far better than those obtained by the more elementary ML techniques, and so provided a more accurate picture of the underlying stimulus-response function.

A total of 205 recordings were collected extracellularly from 68 recording sites in the thalamo-recipient layers of the left primary auditory cortex of anaesthetized rodents (6 CBA/CaJ mice and 4 Long-Evans rats) while a dynamic random chord stimulus (described below) was presented to the right ear. Recordings often reflected the activity of a number of neurons; single neurons were identified by Bayesian spike-sorting techniques [2, 3] whenever possible. The stimulus consisted of 20 ms tone pulses (ramped up and down with a 5 ms cosine gate) presented at random center frequencies, maximal intensities, and times, such that pulses at more than one frequency might be played simultaneously. This stimulus resembled that used in a previous study [4], except in the variation of pulse intensity. The times, frequencies and sound intensities of all tone pulses were chosen independently within the discretizations of those variables (20 ms bins in time, 1/12 octave bins covering either 2–32 or 25–100 kHz in frequency, and 5 dB SPL bins covering 25–70 dB SPL in level). At any time point, the stimulus averaged two tone pulses per octave, with an expected loudness of approximately 73 dB SPL for the 2–32 kHz stimulus and 70 dB SPL for the 25–100 kHz stimulus. Each pulse was ramped up and down with a 5 ms cosine gate. The total duration of each stimulus was 60 s. At each recording site, the 2–32 kHz stimulus was repeated for 20 trials, and the 25–100 kHz stimulus for 10 trials.

Neural responses from all 10 or 20 trials were histogrammed in 20 ms bins aligned with stimulus pulse durations. Thus, in the regression framework, the instantaneous input vector $x_t$ comprised the sound amplitudes at each possible frequency at time $t$, and the output $y_t$ was the number of spikes per trial collected into the $t$th bin. The repetition of the same stimulus made it possible to partition the recorded response power into a stimulus-related (signal) component and a noise component. (For derivation, see Sahani and Linden, "How Linear are Auditory Cortical Responses?", this volume.) Only those 92 recordings in which the signal power was significantly greater than zero were used in this study.

Tests of generalization were performed by cross-validation. The total duration of the stimulus was divided 10 times into a training data segment (9/10 of the total) and a test data segment (1/10), such that all 10 test segments were disjoint. Performance was assessed by the predictive power, that is the test data variance minus average squared prediction error. The 10 estimates of the predictive power were averaged, and normalized by the estimated signal power to give a number less than 1. Note that the predictive power could be negative in cases where the mean was a better description of the test data than was the model prediction. In graphs of the predictive power as a function of noise level, the estimate of the noise power is also shown after normalization by the estimated signal power.

## 3 Evidence optimization for linear regression

As is common in regression problems, it is convenient to collect all the stimulus vectors and observed responses into matrices. Thus, we described the input by a matrix $X$, the $t$th column of which is the input lag-vector $\mathbf{x}_t$. Similarly, we collect the outputs into a row vector $Y$, the $t$th element of which is $y_t$. The first $L-1$ time-steps are dropped to avoid

incomplete lag-vectors. Then, assuming independent noise in each time bin, we combine the individual probabilities to give:

$$
\mathsf{P}\left(Y \mid X, \mathbf{w}, \sigma^2\right) = \frac{1}{\sqrt{|2\pi\sigma^2 I|}} \exp\left(-\frac{1}{2}\frac{(Y - \mathbf{w}^\mathsf{T} X)(Y - \mathbf{w}^\mathsf{T} X)^\mathsf{T}}{\sigma^2}\right) \tag{2}
$$

We now choose the prior distribution on $\mathbf{w}$ to be normal with zero mean (having no prior reason to favour either positive or negative weights) and covariance matrix $C$. Then the joint density of $Y$ and $\mathbf{w}$ is

$$
\mathsf{P}\left(Y, \mathbf{w} \mid X, C, \sigma^2\right) = \frac{1}{Z} \exp\left[-\frac{1}{2}\left(\frac{(Y - \mathbf{w}^\mathsf{T} X)(Y - \mathbf{w}^\mathsf{T} X)^\mathsf{T}}{\sigma^2} - \mathbf{w}^\mathsf{T} C^{-1} \mathbf{w}\right)\right] \tag{3}
$$

where the normalizer $Z = \sqrt{|2\pi\sigma^2 I| \, |2\pi C|}$. Fixing $Y$ to be the observed values, this implies a normal posterior on $\mathbf{w}$ with variance $\Sigma = (\frac{XX^\mathsf{T}}{\sigma^2} + C^{-1})^{-1}$ and mean $\mu = \Sigma\frac{XY^\mathsf{T}}{\sigma^2}$. By integrating this normal density in $\mathbf{w}$ we obtain an expression for the evidence:

$$
\mathcal{E}(C, \sigma^2) = \mathsf{P}\left(Y \mid X, C, \sigma^2\right) = \sqrt{\frac{|2\pi\Sigma|}{|2\pi\sigma^2 I| \, |2\pi C|}} \exp\left[-\frac{1}{2}Y\left(\frac{I}{\sigma^2} - \frac{X^\mathsf{T}\Sigma X}{\sigma^4}\right)Y^\mathsf{T}\right]
$$
$$\tag{4}$$

We seek to optimize this evidence with respect to the hyperparameters in $C$, and the noise variance $\sigma^2$. To do this we need the respective gradients. If the covariance matrix contains a parameter $\theta$, then the derivative of the log-evidence with respect to $\theta$ is given by

$$
\frac{\partial}{\partial\theta}\log\mathcal{E} = \frac{1}{2}\mathrm{Tr}\left[(C - \Sigma - \mu\mu^\mathsf{T})\frac{\partial}{\partial\theta}C^{-1}\right] \tag{5}
$$

while the gradient in the noise variance is

$$
\frac{\partial}{\partial\sigma^2}\log\mathcal{E} = \frac{1}{\sigma^2}\left(-T + \mathrm{Tr}\left[I - \Sigma C^{-1}\right] + \frac{1}{\sigma^2}(Y - \mu^\mathsf{T} X)(Y - \mu^\mathsf{T} X)^\mathsf{T}\right) \tag{6}
$$

where $T$ is the number of training data points.

## 4 Automatic relevance determination (ARD)

The most common evidence optimization scheme for regression is known as automatic relevance determination (ARD). Originally proposed by MacKay and Neal, it has been used extensively in the literature, notably by MacKay[5] and, in a recent application to kernel regression, by Tipping [6]. The prior covariance on the weights is taken to be of the form $C = A^{-1}$ with $A = \mathtt{diag}(\alpha_i)$. That is, the weights are taken to be independent with potentially different prior precisions $\{\alpha_i\}$. Substitution into (5) yields

$$
\frac{\partial}{\partial\alpha_i}\log\mathcal{E} = \frac{1}{2}\left(\alpha_i^{-1} - \Sigma_{ii} - \mu_i^2\right). \tag{7}
$$

Previous authors have noted that, in comparison to simple gradient methods, iteration of fixed point equations derived from this and from (6) converge more rapidly:

$$
\alpha_i^{\mathrm{new}} = \frac{1 - \alpha_i\Sigma_{ii}}{\mu_i^2} \tag{8}
$$

and

$$
(\sigma^2)^{\mathrm{new}} = \frac{(Y - \mu^\mathsf{T} X)(Y - X^\mathsf{T}\mu)}{T - \sum_i(1 - \Sigma_{ii}\alpha_i)} \tag{9}
$$

.

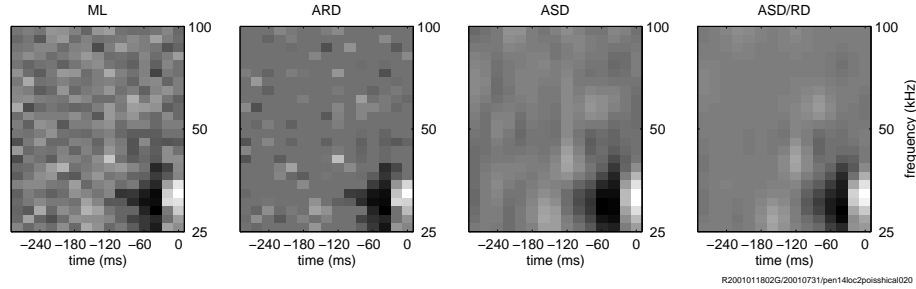

Figure 1: Comparison of various STRF estimates for the same recording.

A pronounced general feature of the maxima discovered by this approach is that many of the optimal precisions are infinite (that is, the variances are zero). Since the prior distribution is centered on zero, this forces the corresponding weight to vanish. In practice, as the iterated value of a precision crosses some pre-determined threshold, the corresponding input dimension is eliminated from the regression problem. The results of evidence optimization suggest that such inputs are irrelevant to predicting the output; hence the name given to this technique. The resulting MAP estimates obtained under the optimized ARD prior thus tend to be sparse, with only a small number of non-zero weights often appearing as isolated spots in the STRF.

The estimated STRFs for one example recording using ML and ARD are shown in the two left-most panels of figure 1 (the other panels show smoothed estimates which will be described below), with the estimated weight vectors rearranged into time-frequency matrices. The sparsity of the ARD solution is evident in the reduction of apparent estimation noise at higher frequencies and longer time lags. This reduction improves the ability of the estimated model to predict novel data by more than a factor of 2 in this case. Assessed by cross-validation, as described above, the ARD estimate accurately predicted 26% of the signal power in test data, whereas the ML estimate (or Wiener kernel) predicted only 12%.

This improvement in predictive quality was evident in every one of the 92 recordings with significant signal power, indicating that the optimized prior does improve estimation accuracy. The left-most panel of figure 2 compares the normalized cross-validation predictive power of the two STRF estimates. The other two panels show the difference in predictive powers as function of noise (in the center) and as a histogram (right). The advantage of the evidence-optimization approach is clearly most pronounced at higher noise levels.

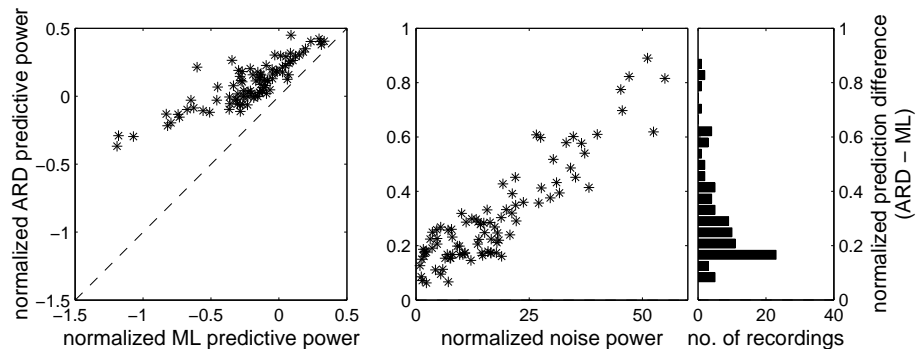

Figure 2: Comparison of ARD and ML predictions.

# 5 Automatic smoothness determination (ASD)

In many regression problems, such as those for which ARD was developed, the different input dimensions are often unrelated; indeed they may be measured in different units. In such contexts, an independent prior on the weights, as in ARD, is reasonable. By contrast, the weights of an STRF are dimensionally and semantically similar. Furthermore, we might expect weights that are nearby in either time or frequency (or space) to be similar in value; that is, the STRF is likely to be smooth on the scale at which we model it.

Here we introduce a new evidence optimization scheme, in which the prior covariance matrix is used to favour smoothing of the STRF weights. The appropriate scale (along either the time or the frequency/space axis) cannot be known *a priori*. Instead, we introduce hyperparameters $\delta_s$ and $\delta_t$ that set the scale of smoothness in the spectral (or spatial) and temporal dimensions respectively, and then, for each recording, optimize the evidence to determine their appropriate values.

The new parameterized covariance matrix, $S$, depends on two $D \times D$ matrices $\Delta_s$ and $\Delta_t$. The $(ij)th$ element of each of these gives the squared distance between the weights $w_i$ and $w_j$ in terms of center frequency (or space) and time respectively. We take

$$S = \exp\left(-\rho - \frac{1}{2}\left(\frac{\Delta_s}{\delta_s^2} + \frac{\Delta_t}{\delta_t^2}\right)\right), \tag{10}$$

where the exponent is taken element by element. In this scheme, the hyperparameters $\delta_s$ and $\delta_t$ set the correlation distances for the weights along the spectral (spatial) and temporal dimensions, while the additional hyperparameter $\rho$ sets their overall scale.

Substitution of (10) into the general hyperparameter derivative expression (5) gives

$$\frac{\partial}{\partial \rho} \log \mathcal{E} = \frac{1}{2}\text{Tr}\left[(S - \Sigma - \mu\mu^\mathsf{T})S^{-1}\right] \tag{11}$$

and

$$\frac{\partial}{\partial \delta_s} \log \mathcal{E} = -\frac{1}{2}\text{Tr}\left[(S - \Sigma - \mu\mu^\mathsf{T})S^{-1}(S \circ \frac{\Delta_s}{\delta_s^3})S^{-1}\right] \tag{12}$$

(where the $\circ$ denotes the Hadamard or Schur product; *i.e.*, the matrices are multiplied element by element), along with a similar expression for $\frac{\partial}{\partial \delta_t} \log \mathcal{E}$. In this case, optimization is performed by simple gradient methods.

The third panel of figure 1 shows the ASD-optimized MAP estimate of the STRF for the same example recording discussed previously. Optimization yielded smoothing width estimates of 0.96 (20 ms) bins in time and 2.57 (1/12 octave) bins in frequency; the effect of this smoothing of the STRF estimate is evident. ASD further improved the ability of the linear kernel to predict test data, accounting for 27.5% of the signal power in this example.

In the population of 92 recordings (figure 3, upper panels) MAP estimates based on the ASD-optimized prior again outperformed ML (Wiener kernel) estimates substantially on every single recording considered, particularly on those with poorer signal-to-noise ratios. They also tended to predict more accurately than the ARD-based estimates (figure 3, lower panels). The improvement over ARD was not quite so pronounced (although it was frequently greater than in the example of figure 1).

# 6 ARD in an ASD-defined basis

The two evidence optimization frameworks presented above appear inconsistent. ARD yields a sparse, independent prior, and often leads to isolated non-zero weights in the estimated STRF. By contrast, ASD is explicitly designed to recover smooth STRF estimates.

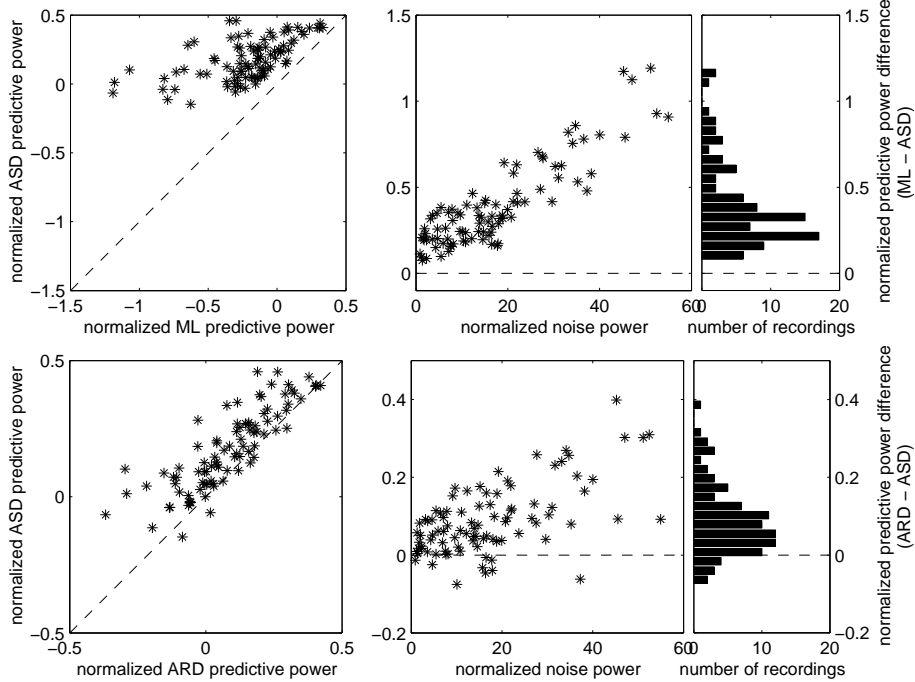

Figure 3: Comparison of ASD predictions to ML and ARD.

Nonetheless, both frameworks appear to improve the ability of estimated models to generalize to novel data. We are thus led to consider ways in which features of both methods may be combined.

By decomposing the prior covariance $C = R^\mathsf{T} R$, it is possible to rewrite the joint density of (3) as

$$\mathsf{P}\left(Y, \mathbf{w} \mid X, C, \sigma^2\right) \propto \exp -\frac{1}{2} \left( \frac{(Y - \mathbf{w}^\mathsf{T} R^{-1} R X)(Y - \mathbf{w}^\mathsf{T} R^{-1} R X)^\mathsf{T}}{\sigma^2} - \mathbf{w}^\mathsf{T} R^{-1} R^{-1\mathsf{T}} \mathbf{w} \right).$$
(13)

Making the substitutions $X' = RX$ and $\mathbf{w}' = R^{-1\mathsf{T}} \mathbf{w}$, this expression may be recognized as the joint density for a transformed regression problem with unit prior covariance (the normalizing constant, not shown, is appropriately transformed by the Jacobean associated with the change in variables). If now we introduce and optimize a diagonal prior covariance of the ARD form in this *transformed* problem, we are indirectly optimizing a covariance matrix of the form $C = R^\mathsf{T} A^{-1} R$ in the original basis. Intuitively, the sparseness driven by ARD is applied to basis vectors drawn from the rows of the transformation matrix $R$, rather than to individual weights. If this basis reflects the smoothness prior obtained from ASD then the resulting prior will combine the smoothness and sparseness of two approaches.

We choose $R$ to be the (positive branch) matrix square root of the optimal prior matrix $S$ (see (10)) obtained from ASD. If the eigenvector decomposition of $S$ is $V D V^\mathsf{T}$, then $R = V D^{1/2} V^\mathsf{T}$, where the diagonal elements of $D^{1/2}$ are the positive square roots of the eigenvalues of $S$. The components of $R$, defined in this way, are Gaussian basis vectors slightly narrower than those in $S$ (this is easily seen by noting that the eigenvalue spectrum for the Toeplitz matrix $S$ is given by the Fourier transform, and that the square-root of the Gaussian function in the Fourier space is a Gaussian of larger width, corresponding to a smaller width in the original space). Thus, weight vectors obtained through ARD

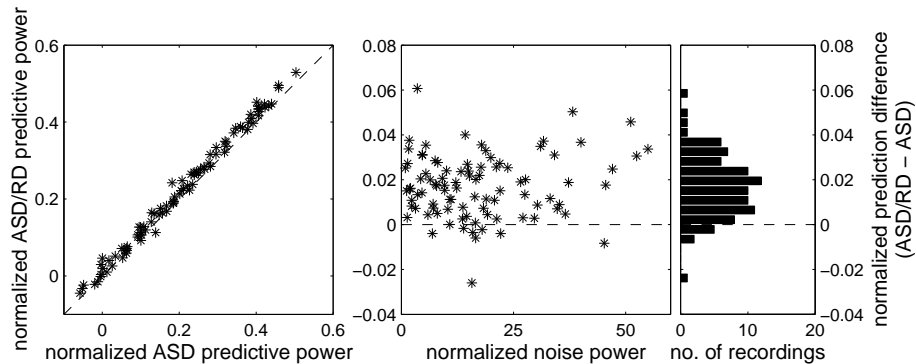

Figure 4: Comparison of ARD in the ASD basis and simple ASD

in this basis will be formed by a superposition of Gaussian components, each of which individually matches the ASD prior on its covariance.

The results of this procedure (labelled ASD/RD) on our example recording are shown in the rightmost panel of figure 1. The combined prior shows a similar degree of smoothing to the ASD-optimized prior alone; in addition, like the ARD prior, it suppresses the apparent background estimation noise at higher frequencies and longer time lags. Predictions made with this estimate are yet more accurate, capturing 30% of the signal power. This improvement over estimates derived from ASD alone is borne out in the whole population (figure 4), although the gain is smaller than in the previous cases.

## 7 Conclusions

We have demonstrated a succession of evidence-optimization techniques which appear to improve the accuracy of STRF estimates from noisy data. The mean improvement in prediction of the ASD/RD method over the Wiener kernel is 40% of the stimulus-related signal power. Considering that the best linear predictor would on average capture no more than 40% of the signal power in these data even in the absence of noise (Sahani and Linden, "How Linear are Auditory Cortical Responses?", this volume), this is a dramatic improvement. These results apply to the case of linear models; our current work is directed toward extensions to non-linear SRFs within an augmented linear regression framework.

## References

[1] Marmarelis, P. Z & Marmarelis, V. Z. (1978) *Analysis of Physiological Systems*. (Plenum Press, New York).

[2] Lewicki, M. S. (1994) *Neural Comp.* **6**, 1005–1030.

[3] Sahani, M. (1999) Ph.D. thesis (California Institute of Technology, Pasadena, CA).

[4] deCharms, R. C, Blake, D. T, & Merzenich, M. M. (1998) *Science* **280**, 1439–1443.

[5] MacKay, D. J. C. (1994) *ASHRAE Transactions* **100**, 1053–1062.

[6] Tipping, M. E. (2001) *J. Machine Learning Res.* **1**, 211–244.
